# Congruence between model and human attention reveals unique signatures of critical visual events

**Robert J. Peters**[*]
Department of Computer Science
University of Southern California
Los Angeles, CA 90089
rjpeters@usc.edu

**Laurent Itti**
Departments of Neuroscience and Computer Science
University of Southern California
Los Angeles, CA 90089
itti@usc.edu

## Abstract

Current computational models of bottom-up and top-down components of attention are predictive of eye movements across a range of stimuli and of simple, fixed visual tasks (such as visual search for a target among distractors). However, to date there exists no computational framework which can reliably mimic human gaze behavior in more complex environments and tasks, such as driving a vehicle through traffic. Here, we develop a hybrid computational/behavioral framework, combining simple models for bottom-up salience and top-down relevance, and looking for changes in the predictive power of these components at different critical event times during 4.7 hours (500,000 video frames) of observers playing car racing and flight combat video games. This approach is motivated by our observation that the predictive strengths of the salience and relevance models exhibit reliable temporal signatures during critical event windows in the task sequence—for example, when the game player directly engages an enemy plane in a flight combat game, the predictive strength of the salience model increases significantly, while that of the relevance model decreases significantly. Our new framework combines these temporal signatures to implement several event detectors. Critically, we find that an event detector based on fused behavioral and stimulus information (in the form of the model's predictive strength) is much stronger than detectors based on behavioral information alone (eye position) or image information alone (model prediction maps). This approach to event detection, based on eye tracking combined with computational models applied to the visual input, may have useful applications as a less-invasive alternative to other event detection approaches based on neural signatures derived from EEG or fMRI recordings.

## 1   Introduction

The human visual system provides an arena in which objects compete for our visual attention, and a given object may win the competition with support from a number of influences. For an example, an moving object in our visual periphery may capture our attention because of its *salience*, or the degree to which it is unusual or surprising given the overall visual scene [1]. On the other hand, a piece of fruit in a tree may capture our attention because of its *relevance* to our current foraging task, in which we expect rewarding items to be found in certain locations relative to tree trunks, and to have particular visual features such as a reddish color [2, 3]. Computational models of each of these influences have been developed and have individually been extensively characterized in terms of their ability to predict an overt measure of attention, namely gaze position [4, 5, 6, 3, 7, 8, 9]. Yet how do the real biological factors modeled by such systems interact in real-world settings [10]? Often salience and relevance are competing factors, and sometimes one factor is so strong that it

---

[*]webpage: http://ilab.usc.edu/rjpeters/

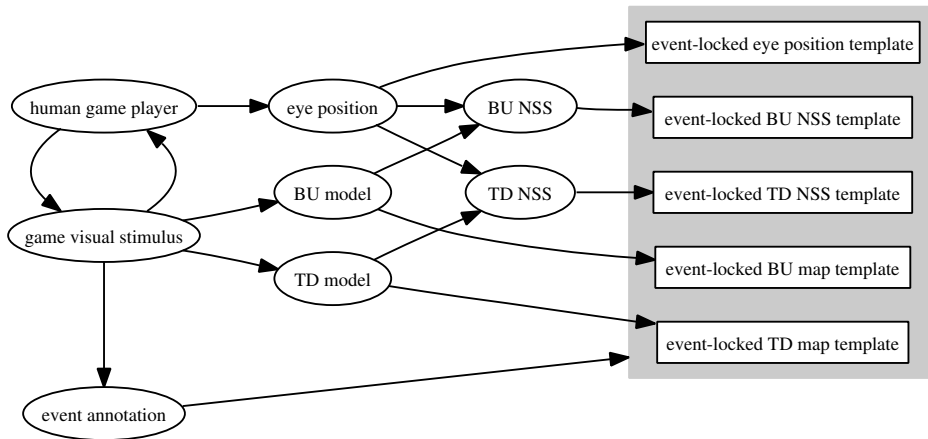

Figure 1: Our computational framework for generating detector templates which can be used to detect key events in video sequences. A human game player interacts with a video game, generating a sequence of video frames from the game, and a sequence of eye position samples from the game player. The video frames feed into computational models for predicting bottom-up (BU) salience and top-down (TD) relevance influences on attention. These predictions are then compared with the observed eye position using a "normalized scanpath saliency" (NSS) metric. Finally, the video game sequence is annotated with key event times, and these are used to generate event-locked templates from each of the game-related signals. These templates are used to try to detect the events in the original game sequences, and the results are quantified with metrics from signal detection theory.

overrides our best efforts to ignore it, as in the case of oculomotor capture [11]. How does the visual system decide which factor dominates, and how does this vary as a function of the current task? We propose that one element of learning sophisticated visual or visuomotor tasks may be learning which attentional influences are important for each phase of the task. A key question is how to build models that can capture the effects of rapidly changing task demands on behavior.

Here we address that question in the context of playing challenging video games, by comparing eye movements recorded during game play with the predictions of a combined salience/relevance computational model. Figure 1 illustrates the overall framework. The important factor in our approach is that we identify key game events (such as destroying an enemy plane, or crashing the car during driving race) which can be used as proxy indicators of likely transitions in the observer's task set. Then we align subsequent analysis on these event times, such that we can detect repeatable changes in model predictive strength within temporal windows around the key events. Indeed, we find significant changes in the predictive strength of both salience and relevance models within these windows, including more than 8-fold increases in predictive strength as well as complete shifts from predictive to anti-predictive behavior. Finally we show that the predictive strength signatures formed in these windows can be used to detect the occurrence of the events themselves.

## 2    Psychophysics and eye tracking

Five subjects (four male, one female) participated under a protocol approved by the Institutional Review Board of the University of Southern California. Subjects played two challenging games on a Nintendo GameCube: Need For Speed Underground (a car racing game) and Top Gun (a flight combat game). All of the subjects had at least some prior experience with playing video games in general, but none of the subjects had prior experience with the particular games involved in our experiment. For each game, subjects first practiced the game for several one-hour sessions on different days until reaching a success criterion (definition follows), and then returned for a one-hour eye tracking session with that game. Within each game, subjects learned to play three game levels, and during eye tracking, each subject played each game level twice. Thus, in total, our recorded data set consists of video frames and eye tracking data from 60 clips (5 subjects × 2 games per subject × 3 levels per game × 2 clips per level) covering 4.7 hours.

**Need For Speed: Underground** (NFSU). In this game, players control a car in a race against three other computer-controlled racers in a three-lap race, with a different race course for each game level. The game display consists of a first-person view, as if the player were looking out the windshield from the driver's seat of the vehicle, with several "heads-up display" elements showing current elapsed time, race position, and vehicle speed, as well as a race course map (see Figure 2 for sample game frames). The game controller joystick is used simply to steer the vehicle, and a pair of controller buttons are used to apply acceleration or brakes. Our "success" criterion for NFSU was finishing the race in third place or better out of the four racers. The main challenge for players was learning to be able to control the vehicle at a high rate of simulated speed (100+ miles per hour) while avoiding crashes with slow-moving non-race traffic and also avoiding the attempts of competing racers to knock the player's vehicle off course. During eye tracking, the average length of an NFSU level was 4.11 minutes, with a range of 3.14–4.89 minutes across the 30 NFSU recordings.

**Top Gun** (TG). In this game, players control a simulated fighter plane with a success criterion of destroying 12 specific enemy targets in 10 minutes or less. The game controller provides a simple set of flight controls: the joystick controls pitch (forward–backward axis) and combined yaw/roll (left–right axis), a pair of buttons controls thrust level up and down, and another button triggers missile firings toward enemy targets. Two onscreen displays aid the players in finding enemy targets: one is a radar map with enemy locations indicated by red triangles, and another is a direction finder running along the bottom screen showing the player's current compass heading along with the headings to each enemy target. Players' challenges during training involved first becoming familiar with the flight controls, and then learning a workable strategy for using the radar and direction finder to efficiently navigate the combat arena. During eye tracking, the average length of a TG level was 5.29 minutes, with a range of 2.96–8.78 minutes across the 30 TG recordings.

**Eye tracking**. Stimuli were presented on a 22" computer monitor at a resolution of 640×480 pixels and refresh rate of 75 Hz. Subjects were seated at a viewing distance of 80 cm and used a chin-rest to stabilize their head position during eye tracking. Video game frames were captured at 29.97Hz from the GameCube using a Linux computer under `SCHED_FIFO` scheduling, which then displayed the captured frames onscreen for the player's viewing and while simultaneously streaming the frames to disk for subsequent processing. Finally, subjects' eye position was recorded at 240Hz with a hardware-based eye-tracking system (ISCAN, Inc.). In total, we obtained roughly 500,000 video game frames and 4,000,000 eye position samples during 4.7 hours of recording.

## 3   Computational attention prediction models

We developed a computational model which uses existing building blocks for bottom-up and top-down components of attention to generate new eye position prediction maps for each of the recorded video game frames. Then, for each frame, we quantified the degree of correspondence between those maps and the actual eye position recorded from the game player. Although the individual models form the underlying computational foundation of our current study, our focus is not on testing their individual validity for predicting eye movements (which has already been established by prior studies), but rather on using them as components of a new model for investigating relationships between task structure and the relative strength of competing influences on visual attention; therefore we provide only a coarse summary of the workings of the models here and refer the reader to original sources for full details.

**Salience**. Bottom-up salience maps were generated using a model based on detecting outliers in space and spatial frequency according to low-level features intensity, color, orientation, flicker and motion [4]. This model has been previously reported to be significantly predictive of eye positions across a range of stimuli and tasks [5, 6, 7, 8].

**Relevance**. Top-down task-relevance maps were generated using a model [9] which is trained to associate low-level "gist" signatures with relevant eye positions (see also [3]). We trained the task-relevance model with a leave-one-out approach: for each of the 60 game clips, the task-relevance model used for testing against that clip was trained on the video frames and eye position samples from the remaining 59 clips.

**Model/human agreement**. For each video game frame, we used the normalized scanpath saliency (NSS) metric [6] to quantify the agreement between the corresponding human eye position and the

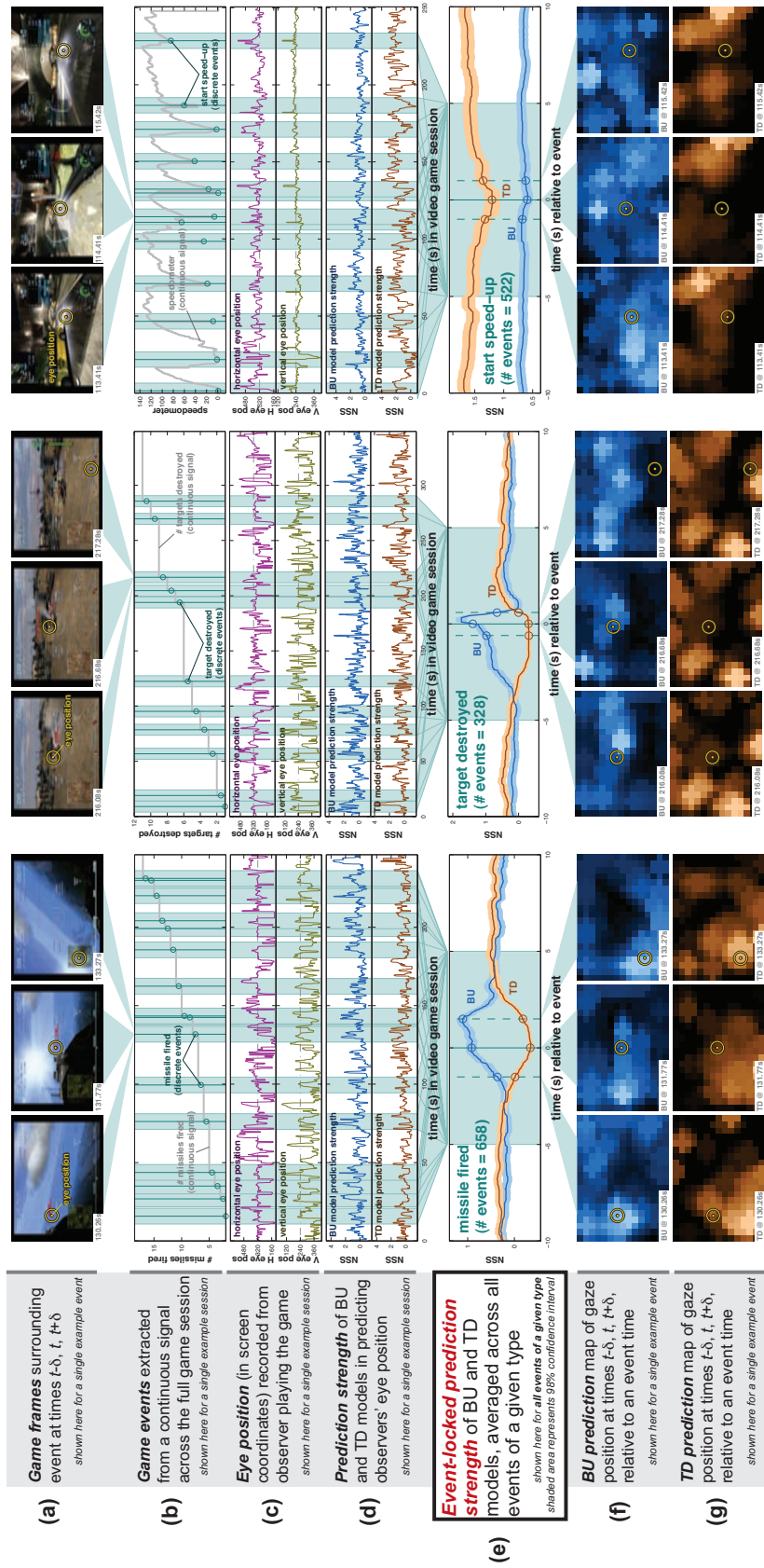

Figure 2: Event-locked analysis of agreement between model-predicted attention maps and observed human eye position. See Section 4 for details.

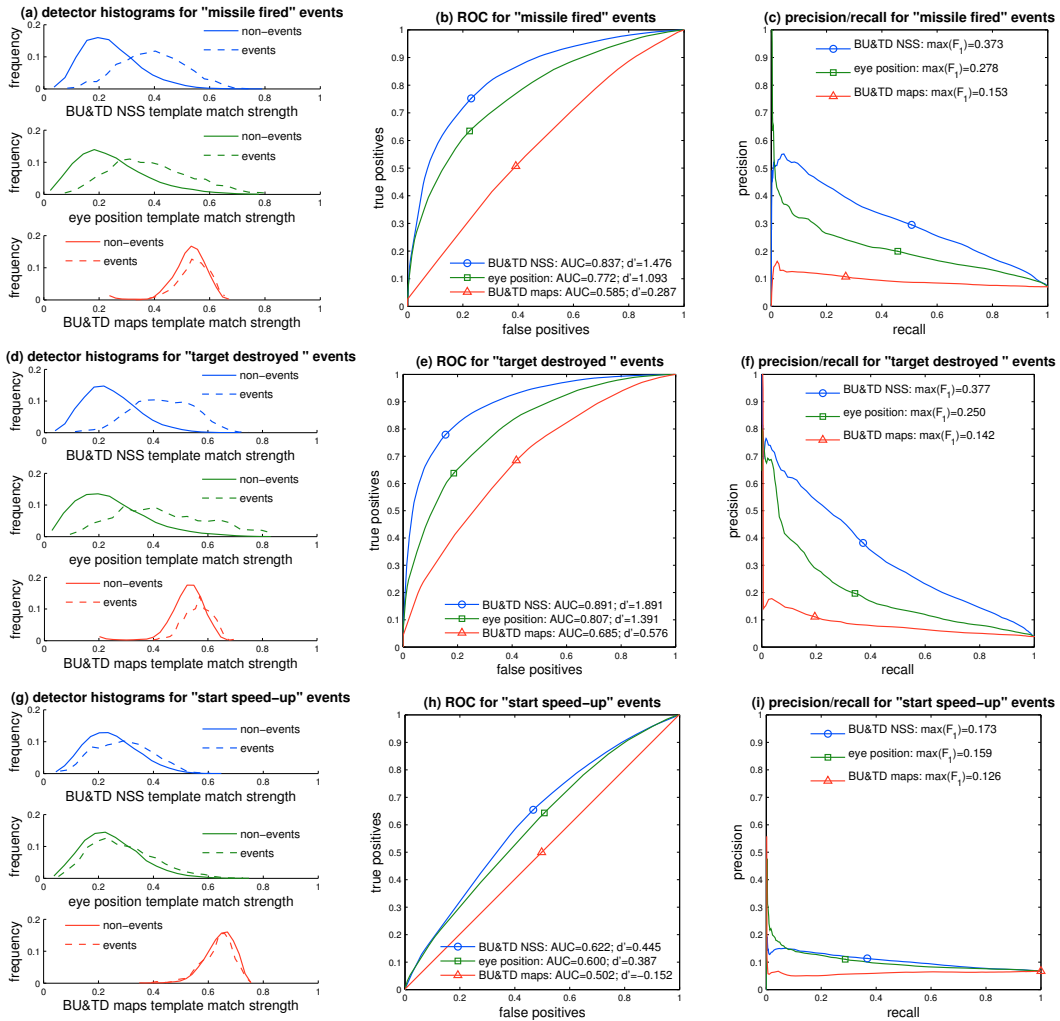

Figure 3: Signal detection results of using event-locked signatures to detect visual events in video game frame sequences. See Section 5 for details.

model maps derived from that frame. Computing the NSS simply involves normalizing the model prediction map to have a mean of zero and a variance of one, and then finding the value in that normalized map at the location of the human eye position. An NSS value of 0 would represent a model at chance in predicting human eye position, while an NSS value of 1 would represent a model for which human eye positions fell at locations with salience (or relevance) one standard deviation above average. Previous studies have typically used the NSS as a summary statistic to describe the predictive strength of a model across an entire sequence of fixations [6, 12]; here, we use it instead as a continuous measure of the instantaneous predictive strength of the models.

## 4 Event-locked analyses

We annotated the video game clips with several pieces of additional information that we could use to identify interesting events (see Figure 2) which would serve as the basis for event-locked analyses. These events were selected on the basis of representing transitions between different task phases. We hypothesized that such events should correlate with changes in the relative strengths of different influences on visual attention, and that we should be able to detect such changes using the previously described models as diagnostic tools. Therefore, after annotating the video clips with the times of each event of interest, we subsequently aligned further analyses on a temporal window of -5s/+5s

around each event (shaded background regions in Figure 2, rows b–e). From those windows we extract the time courses of NSS scores from the salience and relevance models and then compute the average time course across all of the windows, giving an event-locked template showing the NSS signature of that event type (Figure 2e).

*TG "missile fired" events.* In the TG game we looked for times when the player fired missiles (Figure 2, column 1). We selected these events because they represent transitions into a unique task phase, namely the phase of direct engagement with an enemy plane. During most of the TG game playing time, the player's primary task involves actively using the radar and direction finder to locate enemy targets; however, during the time when a missile is in flight the player's only task is to await visual confirmation of the missile destroying its target. Figure 2, column 1, row a illustrates one of the "missile fired" events with captured video frames at -1500ms, 0ms, and +1500ms relative to the event time. Row b uses one of the 30 TG clips to show how the event times represent transitions in a continuous signal (number of missiles fired); a -5s/+5s window around each event is highlighted by the shaded background regions. These windows then propagate through our model-based analysis, where we compare the eye position traces (row c) with the maps predicted by the BU salience (row f) and TD relevance (row g) to generate a continuous sequence of NSS values for each model (row d). Finally, all of the 658 event windows are pooled and we compute the average NSS value along with a 98% confidence interval at each time point in the window, giving event-locked template NSS signatures for the "missile fired" event type (row e). Those signatures show a strong divergence in the predictiveness of the BU and TD models: outside the event window, both models are significantly but weakly predictive of observers' eye positions, with NSS values around 0.3, while inside the event window the BU NSS score increases to an NSS value around 1.0, while the TD NSS score drops below zero for several seconds. We believe this reflects the task phase transition. In general, the TD model has learned that the radar screen and direction finder (toward the bottom left of the game screens) are usually the relevant locations, as illustrated by the elevated activity at those locations in the sample TD maps in row g. Most of the time, that is indeed a good prediction of eye position, reflected by the fact that the TD NSS scores are typically higher than the BU NSS scores outside the event window. However, within the event window, players shift their attention away from the target search task to instead follow the salient objects on the display (enemy target, the missile in flight), which is reflected in the transient upswing in BU NSS scores.

*TG "target destroyed" events.* In the TG game we also considered times when enemy targets were destroyed (Figure 2, column 2). Like the "missile fired" events, these represent transitions between task phases, but whereas the "missile fired" represented transitions from the enemy target search phase into a direct engagement phase, the "target destroyed" events represent the reverse transition; once the player sees that the enemy target has been destroyed, he or she can quickly begin searching the radar and direction finder for the next enemy target to engage. This is reflected in the sample frames shown in Figure 2, column 2, row a, where leading up to the event (at -600ms and 0ms) the player is watching the enemy target, but by +600ms after the event the player has switched back to looking at the direction finder to find a new target. The analysis proceeds as before, using -5s/+5s windows around each of the 328 events to generate average event-locked NSS signatures for the two models (row e). These signatures represent the end of the direct engagement phase whose beginning was represented by the "missile fired" events; here, the BU NSS score reaches an even higher peak of around 1.75 within 50ms after the target being destroyed, and then quickly drops to almost zero by 600ms after the event. Conversely, the TD NSS score is below zero leading up to the event, but then quickly rebounds after the event and transiently goes above its baseline level. Again, we believe these characteristic NSS traces reflect the observer's task transitions.

*NFSU "start speed-up" events.* In the NFSU game, we considered times at which the player just begins recovering from a crash (Figure 2, column 3); players' task is generally to drive as fast as possible while avoiding obstacles, but when players inevitably crash they must transiently shift to a task of trying to recover from the crash. The general driving task typically involves inspecting the horizon line and and focus of expansion for oncoming obstacles, while the crash-recovery task typically involves examining the foreground scene to determine how to get back on course. To automatically identify crash recovery phases, we extracted the speedometer value from each video game frame to form a continuous speedometer history (Figure 2, column 3, row b); we identified "start speed-up" events as upward-turning zero crossings in the acceleration, represented again by shaded background bars in the figure. Again we computed average event-locked NSS signatures for the BU and TD models from -5s/+5s windows around each of the 522 events, giving the traces in

row e. These traces reveal a significant drop in TD NSS scores during the event window, but no significant change in BU NSS scores. The drop in TD NSS scores likely reflects the players' shift of attention away from the usual relevant locations (horizon line, focus of expansion) and toward other regions relevant to the crash-recovery task. However, the lack of change in BU NSS scores indicates that the locations attended during crash recovery where neither more nor less salient than locations attended in general; together, these results suggest that during crash recovery players' attention is more strongly driven by some influence that is not captured well by either of the current BU and TD models.

## 5 Event detectors

Having seen that critical game events are linked with highly significant signatures in the time course of BU and TD model predictiveness, we next asked whether these signatures could be used in turn to predict the events themselves. To test this question, we built event-locked "detector" templates from three sources (see Figure 1): (1) the raw BU and TD prediction maps (which carry explicit information only from the visual input image); (2) the raw eye position traces (which carry explicit information only from the player's behavior); and (3) the BU and TD NSS scores, which represent a fusion of information from the image (BU and TD maps) and from the observer (eye position).

For each of these detector types and for each event type, we compute event-locked signatures just as described in the previous section. For the BU and TD NSS scores, this is exactly what is represented in Figure 2, row e, and for the other two detector types the analysis is analogous. For the BU and TD prediction maps, we compute the event-locked average BU and TD prediction map at each time point within the event window, and for the eye position traces we compute the event-locked average x and y eye position coordinate at each time point. Thus we have signatures for how each of these detector signals is expected to look during the critical event intervals.

Next, we go back to the original detector traces (that is, the raw eye position traces as in Figure 2 row c, or the raw BU and TD maps as in rows f and g, or the raw BU and TD NSS scores as in row d). At each point in those original traces, we compute the correlation coefficient between a temporal window in the trace and the corresponding event-locked detector signature. To combine each pair of correlation coefficients (from BU and TD maps, or from BU and TD NSS, or from x and y eye position) into a single match strength, we scale the individual correlation coefficients to a range of [0...1] and then multiply, to produce a soft logical "and" operation, where both components must have high values in order to produce a high output:

$$\text{BU,TD maps match strength} = r\big(\langle \text{BU}\rangle_{\text{event}}, \text{BU}\big) \cdot r\big(\langle \text{TD}\rangle_{\text{event}}, \text{TD}\big) \tag{1}$$

$$\text{eye position match strength} = r\big(\langle x\rangle_{\text{event}}, x\big) \cdot r\big(\langle y\rangle_{\text{event}}, y\big) \tag{2}$$

$$\text{BU,TD NSS match strength} = r\big(\langle \text{NSS}_{\text{BU}}\rangle_{\text{event}}, \text{NSS}_{\text{BU}}\big) \cdot r\big(\langle \text{NSS}_{\text{TD}}\rangle_{\text{event}}, \text{NSS}_{\text{TD}}\big), \tag{3}$$

where $\langle \cdot \rangle_{\text{event}}$ represents the event-locked template for that signal, and $r(\cdot, \cdot)$ represents the correlation coefficient between the two sequences of values, rescaled from the natural [-1...1] range to a [0...1] range. This yields continuous traces of match strength between the event detector templates and the current signal values, for each video game frame in the data set.

Finally, we adopt a signal detection approach. For each event type, we label every video frame as "during event" if it falls within a -500ms/+500ms window around the event instant, and label it as "during non-event" otherwise. Then we ask how well the match strengths can predict the label, for each of the three detector types (BU and TD maps alone, eye position alone, or BU and TD NSS). Figure 3 shows the results using several signal detection metrics. Each row represents one of the three event types ("missile fired," "target destroyed," and "start speed-up"). The first column (panels a, d, and g) shows the histograms of the match strength values during events and during non-events, for each of the three detector types; this gives a qualitative sense for how well each detector can distinguish events from non-events. The strongest separation between events and non-events is clearly obtained by the BU&TD NSS and eye position detectors for the "missile fired" and "target destroyed" events. Panels b, e, and h show ROC curves for each detector type and event type, along with values for area-under-the-curve (AUC) and d-prime (d'); panels c, f, and i show precision/recall curves with values with for the maximum $F_1$ measure along the curve ($F_1 = (2 \cdot p \cdot r)/(p + r)$, where $p$ and $r$ represent precision and recall). Each metric reflects the same qualitative trends. The highest

scores overall occur for "target destroyed" events, followed by "missile fired" and "start speed-up" events. Within each event type, the highest scores are obtained by the BU&TD NSS detector (representing fused image/behavioral information), followed by the eye position detector (behavioral information only) and then the BU&TD maps detector (image information only).

## 6   Discussion and Conclusion

Our contributions here are twofold: First, we reported several instances in which the degree of correspondence between computational models of attention and human eye position varies systematically as a function of the current task phase. This finding suggests a direct means for integrating low-level computational models of visual attention with higher-level models of general cognition and task performance: the current task state could be linked through a weight matrix to determine the degree to which competing low-level signals may influence overall system behavior.

Second, we reported that variations in the predictive strength of the salience and relevance models are systematic enough that the signals can be used to form template-based detectors of the key game events. Here, the detection is based on signals that represent a fusion of image-derived information (salience/relevance maps) with observer-derived behavior (eye position), and we found that such a combined signal is more powerful than a signal based on image-derived or observer-derived information alone. For event-detection or object-detection applications, this approach may have the advantage of being more generally applicable than a pure computer vision approach (which might require development of algorithms specifically tailored to the object or event of interest), by virtue of its reliance on human/model information fusion. Conversely, the approach of deriving human behavioral information only from eye movements has the advantage of being less invasive and cumbersome than other neurally-based event-detection approaches using EEG or fMRI [13]. Further, although an eye tracker's x/y traces amounts to less raw information than EEG's dozens of leads or fMRI's 10,000s of voxels, the eye-tracking signals also contain a denser and less redundant representation of cognitive information, as they are a manifestation of whole-brain output. Together, these advantages could make our proposed method a useful approach in a number of applications.

## References

[1] L. Itti and P. Baldi. A principled approach to detecting surprising events in video. In *Proc. IEEE Conference on Computer Vision and Pattern Recognition (CVPR)*, pages 631–637, San Siego, CA, Jun 2005.

[2] V. Navalpakkam and L. Itti. Modeling the influence of task on attention. *Vision Research*, 45(2):205–231, January 2005.

[3] A. Torralba, A. Oliva, M.S. Castelhano, and J.M. Henderson. Contextual guidance of eye movements and attention in real-world scenes: the role of global features in object search. *Psychological Review*, 113(4):766–786, October 2006.

[4] L. Itti, C. Koch, and E. Niebur. A model of saliency-based visual attention for rapid scene analysis. *IEEE Transactions on Pattern Analysis and Machine Intelligence*, 20(11):1254–1259, November 1998.

[5] D. Parkhurst, K. Law, and E. Niebur. Modeling the role of salience in the allocation of overt visual attention. *Vision Research*, 42(1):107–123, 2002.

[6] R.J. Peters, A. Iyer, L. Itti, and C. Koch. Components of bottom-up gaze allocation in natural images. *Vision Research*, 45(18):2397–2416, 2005.

[7] R. Carmi and L. Itti. Visual causes versus correlates of attentional selection in dynamic scenes. *Vision Research*, 46(26):4333–4345, Dec 2006.

[8] R.J. Peters and L. Itti. Computational mechanisms for gaze direction in interactive visual environments. In *Proc. ACM Eye Tracking Research and Applications*, pages 27–32, Mar 2006.

[9] R.J. Peters and L. Itti. Beyond bottom-up: Incorporating task-dependent influences into a computational model of spatial attention. In *Proc. IEEE Conference on Computer Vision and Pattern Recognition (CVPR 2007)*, Minneapolis, MN, Jun 2007.

[10] M. Hayhoe and D. Ballard. Eye movements in natural behavior. *Trends in Cognitive Sciences*, 9(4):188–194, April 2005.

[11] J. Theeuwes, A.F. Kramer, S. Hahn, D.E. Irwin, and G.J. Zelinsky. Influence of attentional capture on oculomotor control. *Journal of Experimental Psychology—Human Perception and Performance*, 25(6):1595–1608, December 1999.

[12] W. Einhauser, W. Kruse, K.P. Hoffmann, and P. Konig. Differences of monkey and human overt attention under natural conditions. *Vision Research*, 46(8-9):1194–1209, April 2006.

[13] A.D. Gerson, L.C. Parra, and P. Sajda. Cortically coupled computer vision for rapid image search. *IEEE Transactions on Neural Systems and Rehabilitation Engineering*, 14(2):174–179, June 2006.
